# Reinforcement Learning and Time Perception — a Model of Animal Experiments

**J. L. Shapiro**
Department of Computer Science
University of Manchester
Manchester, M13 9PL U.K.
*jls@cs.man.ac.uk*

**John Wearden**
Department of Psychology
University of Manchester
Manchester, M13 9PL U.K.

## Abstract

Animal data on delayed-reward conditioning experiments shows a striking property — the data for different time intervals collapses into a single curve when the data is scaled by the time interval. This is called the scalar property of interval timing. Here a simple model of a neural clock is presented and shown to give rise to the scalar property. The model is an accumulator consisting of noisy, linear spiking neurons. It is analytically tractable and contains only three parameters. When coupled with reinforcement learning it simulates peak procedure experiments, producing both the scalar property and the pattern of single trial covariances.

## 1 Introduction

An aspect of delayed-reward reinforcement learning problem which has a long history of study in animal experiments, but has been overlooked by theorists, is the learning of the expected time to the reward. In a number of animal experiments, animals need to wait a given time interval after a stimulus before performing an action in order to receive the reward. In order to be able to do this, the animal requires an internal clock or mechanism for perceiving time intervals, as well as a learning system which can tackle more familiar aspects of delayed reward reinforcement learning problem. In this paper it is shown that a simple connectionist model of an accumulator used to measure time duration, coupled to a standard $TD(\lambda)$ reinforcement learning rule reproduces the most prominent features of the animal experiments.

The reason it might be desirable for a learner to learn the expected time to receive a reward is that it allows it to perform the action for an appropriate length of time. An example described by Grossberg and Merrill [4] and modeled in animal experiments by Gibbon and Church [3] is foraging. An animal which had no sense of the typical time to find food might leave too often, thereby spending an inordinate amount of time flying between patches. Alternatively it could remain in a depleted patch and starve. The ability to learn times to rewards is an important aspect of

intelligent behavior more generally.

## 1.1 Peak Procedure Experiments

A typical type of experiment which investigates how animals learn the time between stimulus and reward is the *peak procedure*. In this, the animal is trained to respond after a given time interval $t_r$ has elapsed. Some stimulus (e.g. a light) is presented which stays on during the trial. The animal is able to respond at any time. The animal receives a reward for the first response after the length of time $t_r$. The trial ends when the animal receives the reward.

On some trials, however, no reward is given even when the animal responds appropriately. This is to see when the animal would stop responding. What happens in non-reward trials is that the animal typically will start responding at a certain time, will respond for a period, and then stop responding. Responses averaged over many trials, however, give a smooth curve. The highest response is at the time interval $t_r$, and there is variation around this. The inaccuracy in the response (as measured by the standard deviation in the average response curves for non-reward trials) is also proportional to the time interval. In other words, the ratio of the standard deviation to the mean response time (the *coefficient of variation*) is a constant independent of the time interval.

A more striking property of the timing curves is scalar property, of which the above are two consequences. When the average response rate for non-reward trials is multiplied by the time interval and plotted against the relative time (time divided by the time interval) the data from different time intervals collapse onto one curve.

This strong form of the scalar property can be expressed mathematically as follows. Let $T$ be the actual time since the start of the trial and $\tilde{T}$ be subjective time. Subjective time is the time duration which the animal perceives to have occurred, (or at least appears to perceive judging from its behavior). The experiments show that $\tilde{T}$ varies for a given $T$. This variation can be expressed as a conditional probability, the probability of acting as though the time is $\tilde{T}$ given that the actual time is $T$, which is written $P(\tilde{T}|T)$. The fact that the data collapses implies this probability depends on $T$ and $\tilde{T}$ in a special way,

$$P(\tilde{T}|T) \approx \frac{1}{T} P_{\text{inv}} \left( \frac{\tilde{T}}{T} \right). \tag{1}$$

Here $P_{\text{inv}}$ is the function which describes the shape of the scaled curves. Thus, time acts as a scale factor. This is a strong and striking result. This has been seen in many species, including rats, pigeons, turtles; humans will show similar results if the time intervals are short or if they are prevented from counting through distracting tasks. For reviews of interval timing phenomena, see [5] and [3].

A key question which remains unanswered is: what is the origin of the scalar property. Since the scalar property is ubiquitous, it may be revealing something fundamental about the nature of an internal clock or time perception system. This is especially true if there are only a few known mechanisms which generate this phenomenon. It is well known that any model based on the accumulation of independent errors, such as a clock with a variable pulse-rate, does not produce the scalar property. In such a model it would be the ratio of the *variance* to the mean response time which would be independent of the time interval (a consequence of the law of large numbers). In section 2, a simple stochastic process will be presented which gives rise to scalar timing. In section 3 simulations of the model on the peak

procedure are presented. The model reproduces experimental results on the mean responses and the covariation between responses on non-reward trials.

## 2 The model

### 2.1 An accumulator network of spiking neurons

Here it is shown that a simple connectionist model of an accumulator can give rise to the strong scalar property. The network consists of noisy, linear, spiking neurons which are connected in a random, spatially homogeneous way. The network encodes time as the total activity in the network which grows during the measured time interval. Psychological aspects of the model will be presented elsewhere [8]

The network consists of $N$ identical neurons. The connectivity between neurons is random and defined by a connection matrix $C_{ij}$ which is random and sparse. The connection strength is the same between all connected neurons. An important parameter is the fan-out of the $i$th neuron $C_i$; its average across the network is denoted $C$. Time is in discrete units of size $\tau$, the time required for a spike produced by a neuron to invoke a spike in a connected neuron. There is no refractory period.

The neurons are linear — the expected number of spikes produced by a neuron is $\gamma$ times the number of pre-synaptic spikes. Let $a_i(t)$ denote the number of spikes produced by neuron $i$ at time $t$. This obeys

$$a_i(t+\tau) = \sum_{\alpha=1}^{h_i(t)} \nu_\alpha + I_i(t), \qquad (2)$$

where $h_i(t)$ is the number of spikes feeding into neuron $i$, $h_i(t) = \sum_j C_{ji}x_j(t)$. $I_i(t)$ is the external input at $i$, and $\nu$ is a random variable which determines whether a pre-synaptic spike invokes one in a connected neuron. The mean of $\nu$ is $\gamma$ and the variance is denoted $\sigma_\nu^2$. So the spikes behave independently; saturation effects are ignored. The total activity of the network is

$$n(t) = \sum_{i=1}^{N} a_i(t). \qquad (3)$$

At each time-step, the number of spikes will grow due to the fan-out of the neurons. At the same time, the number of spikes will shrink due to the fact that a spike invokes another spike with a probability less than 1. An essential assumption of this work is that these two processes balance each other, $C\gamma = 1$.

Finally, in order for this network to act as an accumulator, it receives statistically stationary input during the time interval which is being measured, so $I(t)$ is only present during the measured interval and statistically stationary then.

### 2.2 Derivation of the strong scalar property

Here it is shown that the network activity obeys equation (1). Let $y$ be the scaled network activity,

$$y(t) = n(t)/t. \qquad (4)$$

The goal here is the derive the probability distribution for $y$ as a function of time, $P(y|t)$. In order to do this, we use the cumulant generating function (or characteristic function). For any probability distribution, $\rho(x)$, the generating function for

cumulants is,

$$G(s) = \log \left[ \int_\Omega \rho(x) \exp(sx) dx \right] \equiv \sum_{i=0}^\infty \frac{s^i}{i!} \kappa_i \qquad (5)$$

$$(6)$$

where $\Omega$ is the domain of $\rho(x)$, $\kappa_i$ is the $i$th cumulant of $\rho(x)$, and $s$ is just a dummy variable. Taking the $n$th derivative of $G(s)$ with respect to $s$ and setting $s$ to 0 gives $\kappa_i$. Cumulants are like moments, see [1] for some definitions and properties.

We will derive a recursion relation for the cumulant generating function for $y(t)$, denoted $G_y(s;t)$. Let $G_\nu(s)$ denote the generating function for the distribution of $\nu$ and $G_I(s)$ denote the generating functions for the distribution of inputs $I(t)$. These latter two are assumed to be stationary, hence there is no time-dependence. From equation 2 it follows that,

$$G_y(s;t+\tau) = G_I \left( \frac{s}{t+\tau} \right) + \frac{1}{N} \sum_i G_y \left[ tC_i G_\nu \left( \frac{s}{t+\tau} \right); t \right]. \qquad (7)$$

In deriving the above, it was assumed that the activity at each node is statistically the same, and that the fan-out at $i$ is uncorrelated with the activity at $i$ (this requires a sufficiently sparsely connectivity, i.e. no tight loops).

Differentiating the last equation $n$ times with respect to $s$ and setting $s$ to zero produces a set recursion relations for the cumulants of $y$, denoted $\kappa_n$. It is necessary to take terms only up to first order in $1/t$ to find the fixed point distribution. The recursion relations to this order are

$$\kappa_1(t+\tau) = \left(1 - \frac{\tau}{t}\right) \kappa_1(t) + \frac{m_I}{t+\tau} \qquad (8)$$

$$\kappa_n(t+\tau) = \left(1 - n\frac{\tau}{t}\right) \kappa_n(t) + \frac{1}{t} \frac{n(n-1)}{2} C\sigma_\nu^2 \kappa_{n-1}(t)$$

$$+ O\left(\frac{1}{t^2}\right); n > 1. \qquad (9)$$

The above depends upon the mean total input activity $m_I \equiv G_I'(0)$ the average fan-out $C$, and the variance in the noise $\nu$, $\sigma_\nu^2 \equiv G_\nu''(0)$. In general it would depend upon the fan-out times the mean of the noise $\nu$, but that is 1 by assumption. Higher order statistics in $C$ and $\nu$ only contribute to terms which are higher order in $1/t$.

The above equations converge to a fixed point, which shows that $n(t)/t$ has a time-independent distribution for large $t$. The fixed point is found to be

$$G_y(s, \infty) = \sum_{n=0}^\infty \frac{s^n}{n!} \kappa_n(\infty) = \frac{2m_I}{\sigma^2} \log \left(1 - \frac{\sigma_\nu^2}{2\tau} s\right). \qquad (10)$$

Equation 10 is the generating function for a gamma distribution,

$$P_\Gamma(x|a,b) = \frac{\exp(-x/b)x^{a-1}}{b^a \Gamma(a)} \qquad (11)$$

with

$$a = \frac{2m_I}{C\sigma_\nu^2}; \qquad b = \frac{C\sigma_\nu^2}{2\tau}. \qquad (12)$$

Corrections to the fixed point are $O(1/t)$.

What this shows is that for large $t$, the distribution of neural activity, $n$ is scalar,

$$P(n|t) = \frac{1}{t} P_\Gamma \left( \frac{n}{t} | a, b \right); \qquad (13)$$

with $a$ and $b$ defined above.

## 2.3  Reinforcement learning of time intervals

The above model represents a way for a simple connectionist system to measure a time interval. In order to model behavior, the system must learn to association the external stimulus and the clock with the response and the reward. To do this, some additional components are needed.

The $i$th stimulus is represented by a signal $s_i$. The output of the accumulator triggers a set of clock nodes which convert the quantity or activity encoding of time used by the accumulator into a "spatial code" in which particular nodes represent different network activities. This was done because it is difficult to use the accumulator activity directly, as this takes a wide range of values. Each clock node responds to a particular accumulator activity. The output of the $i$th clock node at time $t$ is denoted $X_i(t)$; it is one if the activity is $i$, zero otherwise. It would be more reasonable to use a coarse coding, but this fine-grained encoding is particularly simple. The components of the learning model are shown schematically in figure 1.

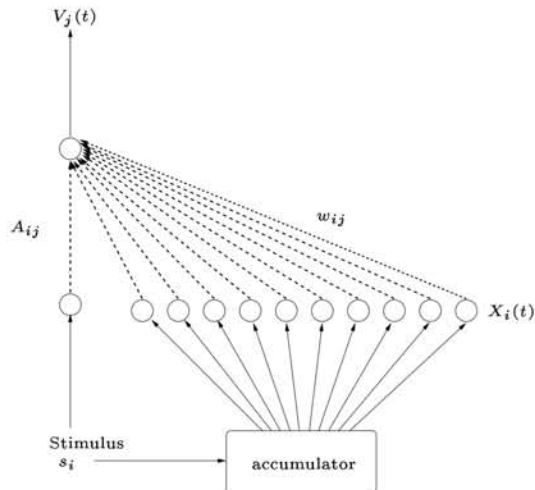

Figure 1: The learning model. The accumulator feeds into a bank of clock nodes, $X_i$, which are tuned to accumulator activities. The response $V_j$ is triggered by simultaneous presence of both the stimulus $s_i$ and the appropriate clock node. Solid lines denote weights which are fixed; dashed lines show weights which learn according to the TD($\lambda$) learning rule.

The stimulus and the clock nodes feed into response nodes. The output of the $j$th response node, $V_j(t)$ is given by

$$V_j(t) = \sum_i w_{ij} X_i(t) + A_{ij} s_i - \theta_j. \tag{14}$$

Here $\theta$ is a threshold, $A_{ij}$ is the association between the stimulus and the response, and $w_{ij}$ is the association between a clock node and the response. Both the stimulus and the appropriate clock node must be present in order for there to be a reasonable probability of a response. The response probability is $V_j(t)$, unless that is negative, in which case there is no response, or is greater than 1, in which case there is definitely a response.

Both $A_{ij}$ and the $w$'s learn via a TD-$\lambda$ learning rule. TD-$\lambda$ is an important learning

rule for modeling associative conditioning; it has been used to model aspects of classical conditioning including Pavlovian conditioning and blocking. For example, a model which is very effective at modeling Pavlovian eye-blink experiments and other classical conditioning results has been proposed by Moore et. al. [6] building on the model of Sutton, Barto, and Desmond (see description in [7]). This model represents time using a tapped delay line; at each time-step, a different node in the delay line is activated. Time acts as one of the conditioned stimuli. The conditioned stimsing temporal difference (TD) reinforcement learning is associated with the response through the unconditioned stimulus. These authors did not attempt to model the scalar property, and in their model time is represented accurately by the system. The model presented here is similar to these models. The clock nodes play the role of the tapped delay-line nodes in that model. However, here they are stimulated by the accumulator rather than each other, and they will follow a stochastic trajectory due to the fluctuating nature of the accumulator

The learning rule for $w_{ij}$ couples to an "eligibility trace" for the clock nodes $\overline{X_i(t)}$ which takes time to build up and decays after the node is turned off. They obey the following equations,

$$\overline{X_i(t+\tau)} = \overline{X_i(t)} + (1 - \gamma\lambda)\left(X_i(t) - \overline{X_i(t)}\right). \tag{15}$$

The standard TD-$\lambda$ learning parameters, $\gamma$ and $\lambda$ are used, see [9]. The learning equations are

$$\Delta w_{ij} = \alpha\delta(t+\tau)\overline{X_i(t)}, \tag{16}$$
$$\Delta A_{ij} = \alpha\delta(t+\tau)s_i, \tag{17}$$
$$\delta(t) = R(t) + \gamma\, V_j(t) - V_j(t-\tau). \tag{18}$$

Here $\alpha$ is a learning rate, $\delta$ is the temporal difference component, $R(t)$ is the reinforcement. The outputs $V_j$ at both times use the current value of the weights. The threshold is set to a constant value ($-1$ in the simulations). It would make no difference if a eligibility trace were used for the stimulus $s_i$, because that was held on during the learning.

## 3  Simulations

The model has been used to simulate peak procedure. In the simulations, the model is forced to respond for the first set of trials (50 trials in the simulations); otherwise the model would never respond. This could represent shaping in real experiments. After that the model learns using reward trials for an additional number of trials (150 trials in these simulations). The system is then run for 1000 trials, every 10th trial is a non-reward trial; the system continues to learn during these trials. Figure 2 shows average over non-reward trials for different time intervals. The scalar property clearly holds.

Gibbon and Church [3] have argued that the covariation between trials is a useful diagnostic to distinguish models of scalar timing. The methodology which they proposed is to fit the results of single non-reward trials from peak procedure experiments to a break-run-break pattern of response The animal is assumed to respond at a low rate until a start time is reached. The animal then responds at a high rate until a stop time is reached, whence it returns to the low response rate. The covariation between the start and stop times between trials is measured and compared to those predicted by theory.

The question Gibbon and Church asked was, how does the start and stop time covary across trials. For example, if the animal starts responding early, does it stop

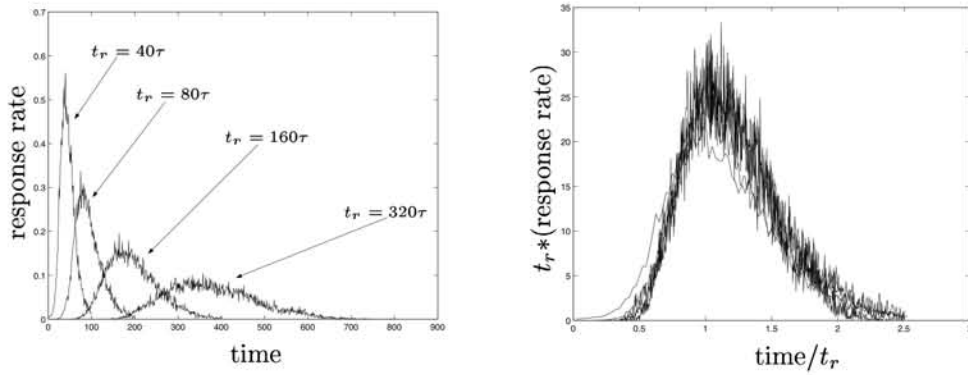

Figure 2: Left) Average response of the spatially encoded network for non-reward trials. The accumulator parameters are: $m_I = 10, C\sigma^2 = 1$ (Poisson limit); learning parameters are $\gamma = 0.75, \lambda = 1$, learning rate $\alpha$ is 0.5. Right) Relative time plotted against response rate times time interval for reinforcement times of $40\tau, 80\tau, 160\tau, 240\tau$, and $320\tau$. All experiments are averages over 100 non-reward trials, which were every 10 trial in 1000 learning trials.

responding early, as though it has a shifted estimate of the time interval? Or does it stop responding late, as though it has a more liberal view about what constitutes the particular interval. The covariance between start and stop parameters addresses this question.

Comparable experiments can be carried out on the model proposed here. The procedure used is described in [2]. Figure 3 shows a comparison with data from reference [2] with simulations. The pattern of covariation found in the simulations is qualitatively similar to that of the animal data. The interesting quantity is the correlation between the start time and the spread (difference between stop and start times). This is negative in both.

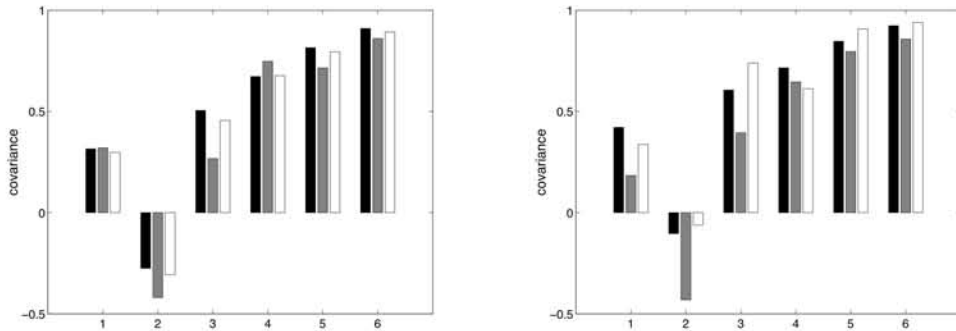

Figure 3: Left) Covariances across individual trials in experiments on rats. Data is taken from Table 2 of reference [2] averaged over the four conditions. The covariances are shown in the following order: 1. start-stop, 2. start-spread, 3. spread-middle, 4. start-middle, 5. stop-spread, 6. stop-middle. The black, gray, and white bars are for times of reinforcement $t_r$ of 15, 30, and 60 seconds respectively. Right) Covariances across individual trials simulated by the model. The reinforcement times are $40\tau, 80\tau$, and $160\tau$. The covariances are given in the same order as in left figure.

# 4 Conclusion

Previous models of interval timing fail to explain its most striking feature — the collapse of the data when scaled by the time interval. We have presented a simple model of an accumulator clock based on spiking, noisy, linear neurons which produces this effect. It is a simple model, analytically tractable, based on a driven branching process. The parameters are: $\tau$ — the time for a spike on one neuron to excite spikes on connected neurons, $m_I$ — the average number of spikes excited externally at each short time interval $\tau$, and the variance of the spike transmission process, which in this model is $\sigma_\nu^2$. A weakness of this model is that it requires fine-tuning of a pair of parameters, so that the expected number of spikes grows in with external excitation only.

Once a scalar clock is produced, simple reinforcement learning can be used to associate the clock signal with appropriate responses. A set of intermediate clock nodes was used to encode time. TD-$\lambda$ reinforcement learning between the intermediate nodes at reinforcement and an eligibility trace simulates peak procedure and the individual trial covariances.

# References

[1] M. Abramowitz and I. A. Stegun, editors. *Handbook of Mathematical Functions*. New York: Dover Publications, 1967.

[2] Russell M. Church, Walter H. Meck, and John Gibbon. Application of scalar timing theory to individual trials. *Journal of Experimental Psychology – Animal Behavior Processes*, 20(2):135–155, 1994.

[3] John Gibbon and Russell M. Church. Representation of time. *Cognition*, 37:23–54, 1990.

[4] Stephen Grossberg and John W. L. Merrill. A neural network model of adaptively timed reinforcement learning and hippocampal dynamics. *Cognitive Brain Research*, 1:3–38, 1992.

[5] S. C. Hinton and W. H. Meck. How time flies: Functional and neural mechansims of interval timing. In C. M. Bradshaw and E. Szadabi, editors, *Time and Behaviour: Psychological and Neurobehavioural Analyses*. Amsterdam: Elsevier Science, 1997.

[6] J. W. Moore, J. E. Desmond, and N. E. Berthier. Adaptively timed conditioned responses and the cerebellum: A neural network approach. *Biological Cybernetics*, 62:17–28, 1989.

[7] John W. Moore, Neil D. Berthier, and Diana E. J. Blazis. Classical eye-blink conditioning: Brain systems and implementation of a computational model. In Michael Gabriel and John Moore, editors, *Learning and Computational Neuroscience: Foundations of Adaptive Networks*, A Bradford Book, pages 359–387. The MIT Press, 1990.

[8] J. L. Shapiro and John Wearden. Modelling scalar timing by an accumulator network of spiking neurons. In preparation, 2001.

[9] Richard S. Sutton and Andrew G. Barto. *Reinforcment Learning: An Introduction*. A Bradford Book. The MIT Press, 1998.
